# Neuromorphic Bistable VLSI Synapses with Spike-Timing-Dependent Plasticity

**Giacomo Indiveri**
Institute of Neuroinformatics
University/ETH Zurich
CH-8057 Zurich, Switzerland
*giacomo@ini.phys.ethz.ch*

## Abstract

We present analog neuromorphic circuits for implementing bistable synapses with spike-timing-dependent plasticity (STDP) properties. In these types of synapses, the short-term dynamics of the synaptic efficacies are governed by the relative timing of the pre- and post-synaptic spikes, while on long time scales the efficacies tend asymptotically to either a potentiated state or to a depressed one. We fabricated a prototype VLSI chip containing a network of integrate and fire neurons interconnected via bistable STDP synapses. Test results from this chip demonstrate the synapse's STDP learning properties, and its long-term bistable characteristics.

## 1 Introduction

Most artificial neural network algorithms based on Hebbian learning use correlations of mean rate signals to increase the synaptic efficacies between connected neurons. To prevent uncontrolled growth of synaptic efficacies, these algorithms usually incorporate also weight normalization constraints, that are often not biophysically realistic. Recently an alternative class of competitive Hebbian learning algorithms has been proposed based on a spike-timing-dependent plasticity (STDP) mechanism [1]. It has been argued that the STDP mechanism can automatically, and in a biologically plausible way, balance the strengths of synaptic efficacies, thus preserving the benefits of both weight normalization and correlation based learning rules [16]. In STDP the precise timing of spikes generated by the neurons play an important role. If a pre-synaptic spike arrives at the synaptic terminal before a post-synaptic spike is emitted, within a critical time window, the synaptic efficacy is increased. Conversely if the post-synaptic spike is emitted soon before the pre-synaptic one arrives, the synaptic efficacy is decreased.

While mean rate Hebbian learning algorithms are difficult to implement using analog circuits, spike-based learning rules map directly onto VLSI [4, 6, 7]. In this paper we present compact analog circuits that, combined with neuromorphic integrate and fire (I&F) neurons and synaptic circuits with realistic dynamics [8, 12, 11] implement STDP learning for short time scales and asymptotically tend to one of two possible states on long time scales. The circuits required to implement STDP, are described in Section 2. The circuits that implement bistability are described in Section 3. The network of I&F neurons used to measure

the properties of the bistable STDP synapse is described in Section 4.

## Long term storage of synaptic efficacies

The circuits that drive the synaptic efficacy to one of two possible states on long time scales, were implemented in order to cope with the problem of long term storage of analog values in CMOS technology. Conventional VLSI capacitors, the devices typically used as memory elements, are not ideal, in that they slowly loose the charge they are supposed to store, due to leakage currents. Several solutions have been proposed for long term storage of synaptic efficacies in analog VLSI neural networks. One of the first suggestions was to use the same method used for dynamic RAM: to periodically *refresh* the stored value. This involves though discretization of the analog value to $N$ discrete levels, a method for comparing the measured voltage to the $N$ levels, and a clocked circuit to periodically refresh the value on the capacitor. An alternative solution is to use analog-to-digital (ADC) converters, an off chip RAM and digital-to-analog converters (DAC), but this approach requires, next to a discretization of the value to $N$ states, bulky ADC and DAC circuits. A more recent suggestion is the one of using *floating gate* devices [5]. These devices can store very precise analog values for an indefinite amount of time using standard CMOS technology [13], but for spike-based learning rules they would require a control circuit (and thus large area) per synapse. To implement dense arrays of neurons with large numbers of dendritic inputs the synaptic circuits should be as compact as possible.

## Bistable synapses

An alternative approach that uses a very small amount of area per synapse is to use bistable synapses. These types of synapses contain minimum feature-size circuits that locally compare the value of the synaptic efficacy stored on the capacitor with a fixed threshold voltage and slowly drive that value either toward a high analog voltage or toward a low one, depending on the output of the comparator (see Section 3).

The assumption that on long time scales the synaptic efficacy can only assume two values is not too severe, for networks of neurons with large numbers of synapses. It has been argued that also biological synapses can be indeed discrete on long time-scales. These assumptions are compatible with experimental data [3] and are supported by experimental evidence [15]. Also from a theoretical perspective it has been shown that the performance of associative networks is not necessarily degraded if the dynamic range of the synaptic efficacy is reduced even to the extreme (two stable states), provided that the transitions between stable states are stochastic [2].

## Related work

Bistable VLSI synapses in networks of I&F neurons have already been proposed in [6], but in those circuits, the synaptic efficacy is always clamped to either a high value or a low one, also for short-term dynamics, as opposed to our case, in which the synaptic efficacy can assume any analog value between the two. In [7] the authors propose a spike-based learning circuit, based on a modified version of Riccati's equation [10], in which the synaptic efficacy is a continuous analog voltage; but their synapses require many more transistors than the solution we propose, and do not incorporate long-term bistability. More recently Bofill and Murray proposed circuits for implementing STDP within a framework of pulse-based neural network circuits [4]. But, next to missing the long-term bistability properties, their synaptic circuits require digital control signals that cannot be easily generated within the framework of neuromorphic networks of I&F neurons [8, 12].

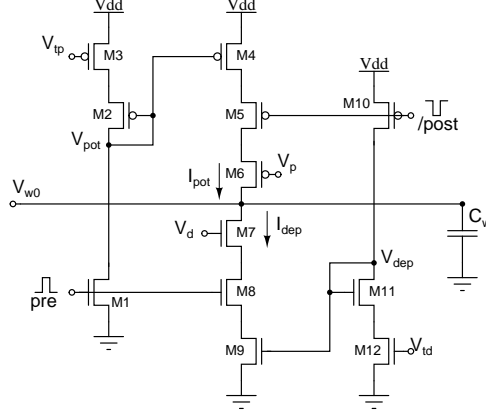

Figure 1: Synaptic efficacy STDP circuit.

## 2 The STDP circuits

The circuit required to implement STDP in a network of I&F neurons is shown in Fig. 1. This circuit increases or decreases the analog voltage $V_{w0}$, depending on the relative timing of the pulses $pre$ and $/post$. The voltage $V_{w0}$ is then used to set the strength of synaptic circuits with realistic dynamics, of the type described in [11]. The pre- and post-synaptic pulses $pre$ and $/post$ are generated by compact, low power I&F neurons, of the type described in [9].

The circuit of Fig. 1 is fully symmetric: upon the arrival of a pre-synaptic pulse $pre$ a waveform $V_{pot}(t)$ (for potentiating $V_{w0}$) is generated. Similarly, upon the arrival of a post-synaptic pulse $/post$, a complementary waveform $V_{dep}(t)$ (for depotentiating $V_{w0}$) is generated. Both waveforms have a sharp onset and decay linearly with time, at a rate set respectively by $V_{tp}$ and $V_{td}$. The pre- and post-synaptic pulses are also used to switch on two gates ($M8$ and $M5$), that allow the currents $I_{dep}$ and $I_{pot}$ to flow, as long as the pulses are high, either increasing or decreasing the weight. The bias voltages $V_p$ on transistor $M6$ and $V_d$ on $M7$ set an upper bound for the maximum amount of current that can be injected into or removed from the capacitor $C_w$. If transistors $M4-M9$ operate in the subthreshold regime [13], we can compute the analytical expression of $I_{pot}(t)$ and $I_{dep}(t)$:

$$I_{pot}(t) = \frac{I_0}{e^{-\frac{\kappa}{U_T}V_{pot}(t-t_{pre})} + e^{-\frac{\kappa}{U_T}V_p}} \tag{1}$$

$$I_{dep}(t) = \frac{I_0}{e^{-\frac{\kappa}{U_T}V_{dep}(t-t_{post})} + e^{-\frac{\kappa}{U_T}V_d}} \tag{2}$$

where $t_{pre}$ and $t_{post}$ are the times at which the pre-synaptic and post-synaptic spikes are emitted, $U_T$ is the thermal voltage, and $\kappa$ is the subthreshold slope factor [13]. The change in synaptic efficacy is then:

$$\begin{cases} \Delta V_{w0} = \frac{I_{pot}(t_{post})}{C_p}\Delta t_{spk} & \text{if } t_{pre} < t_{post} \\ \Delta V_{w0} = -\frac{I_{dep}(t_{pre})}{C_d}\Delta t_{spk} & \text{if } t_{post} < t_{pre} \end{cases} \tag{3}$$

where $\Delta t_{spk}$ is the pre- and post-synaptic spike width, $C_p$ is the parasitic capacitance of node $V_{pot}$ and $C_d$ the one of node $V_{dep}$ (not shown in Fig. 1).

In Fig. 2(a) we plot experimental data showing how $\Delta V_{w0}$ changes as a function of $\Delta t = t_{pre} - t_{post}$ for different values of $V_{td}$ and $V_{tp}$. Similarly, in Fig. 2(b) we show plots

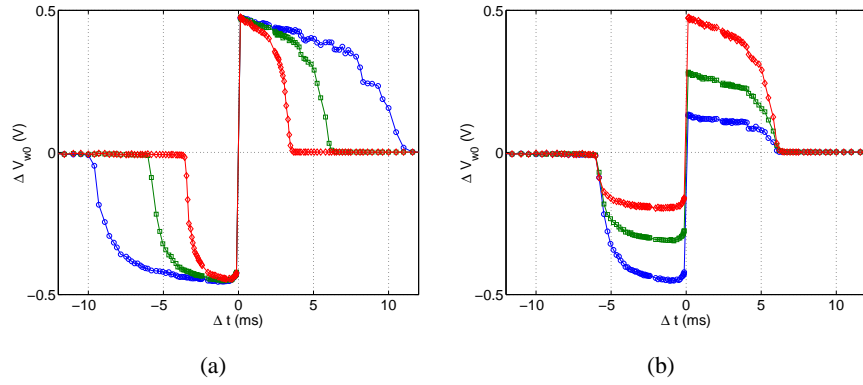

(a)                                                          (b)

Figure 2: Changes in synaptic efficacy, as a function of the difference between pre- and post-synaptic spike emission times $\Delta t = t_{pre} - t_{post}$. (a) Curves obtained for four different values of $V_{pot}$ (in the left quadrant) and four different values of $V_{dep}$ (in the right quadrant). (b) Typical STDP plot, obtained by setting $V_p$ to 4.0V and $V_d$ to 0.6V.

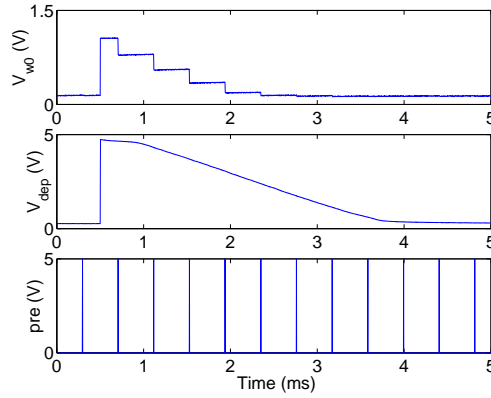

Figure 3: Changes in $V_{w0}$, in response to a sequence of pre-synaptic spikes (top trace). The middle trace shows how the signal $V_{dep}$, triggered by the post-synaptic neuron, decreases linearly with time. The bottom trace shows the series of digital pulses $pre$, generated with every pre-synaptic spike.

of $\Delta V_{w0}$ versus $\Delta t$ for three different values of $V_p$ and three different values of $V_d$. As there are four independent control biases, it is possible to set the maximum amplitude and temporal window of influence independently for positive and negative changes in $V_{w0}$.

The data of Fig. 2 was obtained using a paired-pulse protocol similar to the one used in physiological experiments [14]: one single pair of pre- and post-synaptic spikes was used to measure each $\Delta V_{w0}$ data point, by systematically changing the delay $t_{pre} - t_{post}$ and by separating each stimulation session by a few hundreds of milliseconds (to allow the signals to return to their resting steady-state). Unlike the biological experiments, in our VLSI setup it is possible to evaluate the effect of multiple pulses on the synaptic efficacy, for very long successive stimulation sessions, monitoring all the internal state variables and signals involved in the process. In Fig. 3 we show the effect of multiple pre-synaptic spikes, succeeding a post-synaptic one, plotting a trace of the voltage $V_{w0}$, together with the

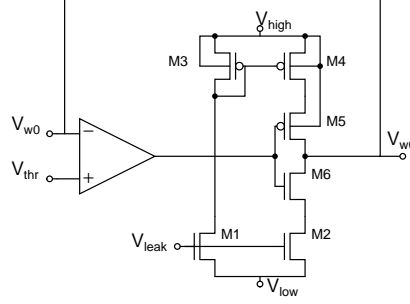

Figure 4: Bistability circuit. Depending on $V_{w0} - V_{thr}$, the comparator drives $V_{w0}$ to either $V_{high}$ or $V_{low}$. The rate at which the circuit drives $V_{w0}$ toward the asymptote is controlled by $V_{leak}$ and imposed by transistors $M2$ and $M4$.

"internal" signal $V_{dep}$, generated by the post-synaptic spike, and the pulses $pre$, generated by the per-synaptic neuron. Note how the change in $V_{w0}$ is a positive one, when the post-synaptic spike follows a pre-synaptic one, at $t = 0.5$ms, and is negative when a series of pre-synaptic spikes follows the post-synaptic one. The effect of subsequent $pre$ pulses following the first post-/pre-synaptic pair is additive, and decreases with time as in Fig. 2. As expected, the anti-causal relationship between pre- and post-synaptic neurons has the net effect of decreasing the synaptic efficacy.

## 3 The bistability circuit

The bistability circuit, shown in Fig. 4, drives the voltage $V_{w0}$ toward one of two possible states: $V_{high}$ (if $V_{w0} > V_{thr}$), or $V_{low}$ (if $V_{w0} < V_{thr}$). The signal $V_{thr}$ is a threshold voltage that can be set externally. The circuit comprises a comparator, and a mixed-mode analog-digital leakage circuit. The comparator is a five transistor transconductance amplifier [13] that can be designed using minimum feature-size transistors. The leakage circuit contains two gates that act as digital switches ($M5, M6$) and four transistors that set the two stable state asymptotes $V_{high}$ and $V_{low}$ and that, together with the bias voltage $V_{leak}$, determine the rate at which $V_{w0}$ approaches the asymptotes. The bistability circuit drives $V_{w0}$ in two different ways, depending on how large is the distance between the value of $V_{w0}$ itself and the asymptote. If $|V_{w0} - V_{as}| > 4U_T$ the bistability circuit drives $V_{w0}$ toward $V_{as}$ linearly, where $V_{as}$ represents either $V_{low}$ or $V_{high}$, depending on the sign of $(V_{w0} - V_{thr})$:

$$\begin{cases} V_{w0}(t) = V_{w0}(0) + \frac{I_{leak}}{C_w}t & \text{if } V_{w0} > V_{thr} \\ V_{w0}(t) = V_{w0}(0) - \frac{I_{leak}}{C_w}t & \text{if } V_{w0} < V_{thr} \end{cases} \tag{4}$$

where $C_w$ is the capacitor of Fig. 1 and

$$I_{leak} = I_0 e^{\frac{\kappa V_{leak} - V_{low}}{U_T}}$$

As $V_{w0}$ gets close to the asymptote and $|V_{w0} - V_{as}| < 4U_T$, transistors $M2$ or $M4$ of Fig. 4 go out of saturation and $V_{w0}$ begins to approach the asymptote exponentially:

$$\begin{cases} V_{w0}(t) = V_{high} - V_{w0}(0)e^{-\frac{I_{leak}}{C_w U_T}t} & \text{if } V_{w0} > V_{thr} \\ V_{w0}(t) = V_{low} + V_{w0}(0)e^{-\frac{I_{leak}}{C_w U_T}t} & \text{if } V_{w0} < V_{thr} \end{cases} \tag{5}$$

On long time scales the dynamics of $V_{w0}$ are governed by the bistability circuit, while on short time-scales they are governed by the STDP circuits and the precise timing of pre- and

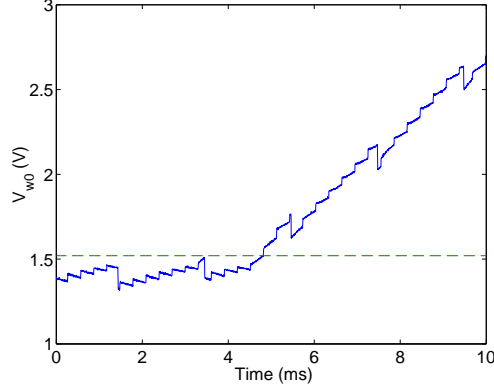

Figure 5: Synaptic efficacy bistability. Transition of $V_{w0}$ from below threshold to above threshold ($V_{thr} = 1.52V$), with leakage rate set by $V_{leak} = 0.25$V and pre- and post-synaptic neurons stimulated in a way to increase $V_{w0}$.

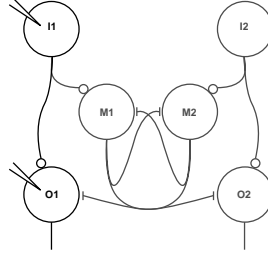

Figure 6: Network of leaky I&F neurons with bistable STDP excitatory synapses and inhibitory synapses. The large circles symbolize I&F neurons, the small empty ones bistable STDP excitatory synapses, and the small bars non-plastic inhibitory synapses. The arrows in the circles indicate the possibility to inject current from an external source, to stimulate the neurons.

post-synaptic spikes. If the STDP short-term dynamics drive $V_{w0}$ above threshold we say that long-term potentiation (LTP) had been induced. And if the short-term dynamics drive $V_{w0}$ below threshold, we say that long-term depression (LTD) has been induced.

In Fig. 5 we show how the synaptic efficacy $V_{w0}$ changes upon induction of LTP, while stimulating the pre- and post-synaptic neurons with uniformly distributed spike trains. The asymptote $V_{low}$ was set to zero, and $V_{high}$ to 2.75V. The pre- and post-synaptic neurons were injected with constant DC currents in a way to increase $V_{w0}$, on average. As shown, the two asymptotes $V_{low}$ and $V_{high}$ act as two attractors, or stable equilibrium points, whereas the threshold voltage $V_{thr}$ acts as an unstable equilibrium point. If the synaptic efficacy is below threshold the short-term dynamics have to fight against the long-term bistability effect, to increase $V_{w0}$. But as soon as $V_{w0}$ crosses the threshold, the bistability circuit switches, the effects of the short-term dynamics are reinforced by the asymptotic drive, and $V_{w0}$ is quickly driven toward $V_{high}$.

## 4  A network of integrate and fire neurons

The prototype chip that we used to test the bistable STDP circuits presented in this paper, contains a symmetric network of leaky I&F neurons [9] (see Fig. 6). The experimental data

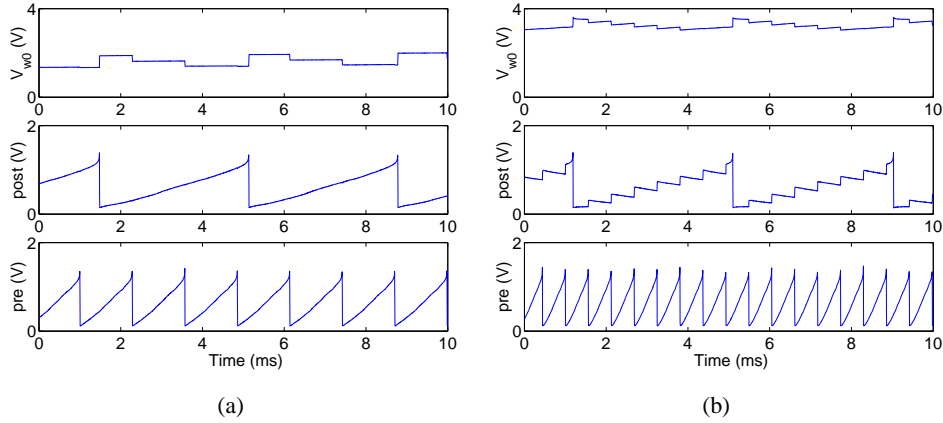

Figure 7: Membrane potentials of pre- and post-synaptic neurons (bottom and middle traces respectively) and synaptic efficacy values (top traces). (a) Changes in $V_{w0}$ for low synaptic efficacy values ($V_{high} = 2.1\text{V}$) and no bistability leakage currents ($V_{leak} = 0$). (b) Changes in $V_{w0}$ for high synaptic efficacy values ($V_{wh} = 3.6V$) and with bistability asymptotic drive ($V_{leak} = 0.25\text{V}$).

of Figs. 2, 3, and 5 was obtained by injecting currents in the neurons labeled I1 and O1 and by measuring the signals from the excitatory synapse on O1. In Fig. 7 we show the membrane potential of I1, O1, and the synaptic efficacy $V_{w0}$ of the corresponding synapse, in two different conditions. Figure 7(a) shows the changes in $V_{w0}$ when both neurons are stimulated but no asymptotic drive is used. As shown $V_{w0}$ strongly depends on the spike patterns of the pre- and post-synaptic neurons. Figure 7(b) shows a scenario in which only neuron I1 is stimulated, but in which the weight $V_{w0}$ is close to its high asymptote ($V_{high} = 3.6\text{V}$) and in which there is a long-term asymptotic drive ($V_{leak} = 0.25$). Even though the synaptic weight stays always in its potentiated state, the firing rate of O1 is not as regular as the one of its efferent neuron. This is mainly due to the small variations of $V_{w0}$ induced by the STDP circuit.

## 5 Discussion and future work

The STDP circuits presented here introduce a source of variability in the spike timing of the I&F neurons that could be exploited for creating VLSI networks of neurons with stochastic dynamics and for implementing spike-based stochastic learning mechanisms [2]. These mechanisms rely on the variability of the input signals (*e.g.* of Poisson distributed spike trains) and on their precise spike-timing in order to induce LTP or LTD only to a small specific sub-set of the synapses stimulated. In future experiments we will characterize the properties of the bistable STDP synapse in response to Poisson distributed spike trains, and measure transition probabilities as functions of input statistics and circuit parameters.

We presented compact neuromorphic circuits for implementing bistable STDP synapses in VLSI networks of I&F neurons, and showed data from a prototype chip. We demonstrated how these types of synapses can either store their LTP or LTD state for long-term, or switch state depending on the precise timing of the pre- and post-synaptic spikes. In the near future, we plan to use the simple network of I&F neurons of Fig. 6, present on the prototype chip, to analyze the effect of bistable STDP plasticity at a network level. On the long term,

we plan to design a larger chip with these circuits to implement a re-configurable network of I&F neurons of O(100) neurons and O(1000) synapses, and use it as a real-time tool for investigating the computational properties of competitive networks and selective attention models.

**Acknowledgments**

I am grateful to Rodney Douglas and Kevan Martin for their support, and to Shih-Chii Liu and Stefano Fusi for constructive comments on the manuscript. Some of the ideas that led to the design and implementation of the circuits presented were inspired by the *Telluride Workshop on Neuromorphic Engineering* (http://www.ini.unizh.ch/telluride).

# References

[1] L. F. Abbott and S. Song. Asymmetric hebbian learning, spike liming and neural response variability. In *Advances in Neural Information Processing Systems*, volume 11, pages 69–75, 1998.

[2] D. J. Amit and S. Fusi. Dynamic learning in neural networks with material synapses. *Neural Computation*, 6:957, 1994.

[3] T. V. P. Bliss and G. L. Collingridge. A synaptic model of memory: Long term potentiation in the hippocampus. *Nature*, 31:361, 1993.

[4] A. Bofill and A.F. Murray. Circuits for VLSI implementation of temporally asymmetric Hebbian learning. In T. G. Dietterich, S. Becker, and Z. Ghahramani, editors, *Advances in Neural Information processing systems*, volume 14. MIT Press, Cambridge, MA, 2001.

[5] C. Diorio, P. Hasler, B.A. Minch, and C. Mead. A single-transistor silicon synapse. *IEEE Trans. Electron Devices*, 43(11):1972–1980, 1996.

[6] S. Fusi, M. Annunziato, D. Badoni, A. Salamon, and D.J. Amit. Spike-driven synaptic plasticity: theory, simulation, VLSI implementation. *Neural Computation*, 12:2227–2258, 2000.

[7] P. Häfiger, M. Mahowald, and L. Watts. A spike based learning neuron in analog VLSI. In M. C. Mozer, M. I. Jordan, and T. Petsche, editors, *Advances in neuralinformation processing systems*, volume 9, pages 692–698. MIT Press, 1997.

[8] G. Indiveri. Modeling selective attention using a neuromorphic analog VLSI device. *Neural Computation*, 12(12):2857–2880, December 2000.

[9] G. Indiveri. A low-power adaptive integrate-and-fi re neuron circuit. In *ISCAS 2003. The 2003 IEEE International Symposium on Circuits and Systems, 2003*. IEEE, 2003.

[10] T. Kohonen. *Self-Organization and Associative Memory*. Springer Series in Information Sciences. Springer Verlag, 2nd edition, 1988.

[11] S.-C. Liu, M. Boegerhausen, and S. Pascal. Circuit model of short-term synaptic dynamics. In *Advances in Neural Information Processing Systems*, volume 15, Cambridge, MA, December 2002. MIT Press.

[12] S.-C. Liu, J. Kramer, G. Indiveri, T. Delbruck, T. Burg, and R. Douglas. Orientation-selective aVLSI spiking neurons. *Neural Networks*, 14(6/7):629–643, 2001. Special Issue on Spiking Neurons in Neuroscience and Technology.

[13] S.-C. Liu, J. Kramer, G. Indiveri, T. Delbruck, and R. Douglas. *Analog VLSI:Circuits and Principles*. MIT Press, 2002.

[14] H. Markram, J. Lübke, M. Frotscher, and B. Sakmann. Regulation of synaptic effi cacy by coincidence of postsynaptic APs and EPSPs. *Science*, 275:213–215, 1997.

[15] C. C. H. Petersen, R. C. Malenka, R. A. Nicoll, and J. J. Hopfield. All-ornone potentiation at CA3-CA1 synapses. *Proc. Natl. Acad. Sci.*, 95:4732, 1998.

[16] S. Song, K. D. Miller, and L. F. Abbot. Competitive Hebbian learning through spike-timing-dependent plasticity. *Nature Neuroscience*, 3(9):919–926, 2000.
